# Selective Prediction of Financial Trends with Hidden Markov Models

**Ran El-Yaniv and Dmitry Pidan**
Department of Computer Science, Technion
Haifa, 32000 Israel
{rani,pidan}@cs.technion.ac.il

## Abstract

Focusing on short term trend prediction in a financial context, we consider the problem of selective prediction whereby the predictor can abstain from prediction in order to improve performance. We examine two types of selective mechanisms for HMM predictors. The first is a rejection in the spirit of Chow's well-known ambiguity principle. The second is a specialized mechanism for HMMs that identifies low quality HMM states and abstain from prediction in those states. We call this model *selective HMM (sHMM)*. In both approaches we can trade-off prediction coverage to gain better accuracy in a controlled manner. We compare performance of the ambiguity-based rejection technique with that of the sHMM approach. Our results indicate that both methods are effective, and that the sHMM model is superior.

## 1   Introduction

*Selective prediction* is the study of predictive models that can automatically qualify their own predictions and output "don't know" when they are not sufficiently confident. Currently, manifestations of selective prediction within machine learning mainly exist in the realm of inductive classification, where this notion is often termed 'classification with a reject option.' In the study of a reject option, which was initiated more than 40 years ago by Chow [5], the goal is to enhance accuracy (or reduce 'risk') by compromising the coverage. For a classifier or predictor equipped with a rejection mechanism we can quantify its performance profile by evaluating its *risk-coverage (RC) curve*, giving the functional relation between error and coverage. The RC curve represents a trade-off: the more coverage we compromise, the more accurate we can expect to be, up to the point where we reject everything and (trivially) never err. The essence of selective classification is to construct classifiers achieving useful (and optimal) RC trade-offs, thus providing the user with *control* over the choice of desired risk (with its associated coverage compromise).

Our longer term goal is to study selective prediction models for general sequential prediction tasks. While this topic has only been sparsely considered in the literature, we believe that it has great potential in dealing with difficult problems. As a starting point, however, in this paper we focus on the restricted objective of predicting next-day trends in financial sequences. While limited in scope, this problem serves as a good representative of difficult sequential data [17]. A very convenient and quite versatile modeling technique for analyzing sequences is the Hidden Markov Model (HMM). Therefore, the goal we set had been to introduce selection mechanisms for HMMs, capable of achieving useful risk-coverage trade-off in predicting next-day trends.

To this end we examined two approaches. The first is a straightforward application of Chow's ambiguity principle implemented with HMMs. The second is a novel and specialized technique utilizing the HMM state structure. In this approach we identify latent states whose prediction quality is systematically inferior, and abstain from predictions while the underlying source is likely to be in

those states. We call this model *selective HMM (sHMM)*. While this natural approach can work in principle, if the HMM does not contain sufficiently many "fine grained" states, whose probabilistic volume (or "visit rate") is small, the resulting risk-coverage trade-off curve will be a coarse step function that will prevent fine control and usability. One of our contributions is a solution to this coarseness problem by introducing algorithms for refining sHMMs. The resulting refined sHMMs give rise to smooth RC trade-off curves.

We present the results of quite extensive empirical study showing the effectiveness of our methods, which can increase the edge in predicting next-day trends. We also show the advantage of sHMMs over the classical Chow approach.

## 2 Preliminaries

### 2.1 Hidden Markov Models in brief

A Hidden Markov Model (HMM) is a generative probabilistic state machine with latent states, in which state transitions and observations emissions represent first-order Markov processes. Given an observation sequence, $O = O_1, \ldots, O_T$, hypothesized to be generated by such a model, we would like to "reverse engineer" the most likely (in a Bayesian sense) state machine giving rise to $O$, with associated latent state sequence $S = S_1, \ldots, S_T$. An HMM is defined as $\lambda \triangleq \langle Q, M, \pi, A, B \rangle$, where $Q$ is a set of states, $M$ is the number of observations, $\pi$ is the initial states distribution, $\pi_i \triangleq \mathbb{P}[S_1 = q_i]$, $A = (a_{ij})$ is the transition matrix, $a_{ij} \triangleq \mathbb{P}[S_{t+1} = q_j \mid S_t = q_i]$, and $B = (b_j(k))$ is the observation emission matrix, $b_j(k) \triangleq \mathbb{P}[O_t = v_k \mid S_t = q_j]$.

Given an HMM $\lambda$ and observation sequence $O$, an efficient algorithm for calculating $\mathbb{P}[O \mid \lambda]$ is the *forward-backward procedure* (see details in, e.g., Rabiner [16]). The estimation of the HMM parameters (training) is traditionally performed using a specialized expectation-maximization (EM) algorithm called the *Baum-Welch algorithm* [2]. For a large variety of problems it is also essential to identify the "most likely" state sequence associated with a given observation sequence. This is commonly accomplished using the *Viterbi algorithm* [22], which computes $\arg\max_S \mathbb{P}[S \mid O, \lambda]$. Similarly, one can identify the most likely "individual" state, $\arg\max_q \mathbb{P}[S_t = q \mid O, \lambda]$, corresponding to time $t$.

### 2.2 Selective Prediction and the RC Trade-off

To define the performance parameters in selective prediction we utilize the following definitions for selective classifiers from [6, 7]. A selective (binary) classifier is represented as a pair of functions $\langle f, g \rangle$, where $f$ is a binary classifier and $g : X \rightarrow \{0, 1\}$ is a binary qualifier for $f$: whenever $g(x) = 1$, the prediction $f(x)$ is accepted, and otherwise it is ignored. The performance of a selective classifier is measured by its *coverage* and *risk*. Coverage is the expected volume of non-rejected data instances, $C \triangleq \mathbb{E}[g(X)]$, (where expectation is w.r.t. the unknown underlying distribution) and the risk is the error rate over non-rejected instances, $R \triangleq \mathbb{E}[\mathbb{I}(f(X) \neq Y)g(X)] / C$, where $Y$ represents the true classification.

The purpose of a selective prediction model is to provide "sufficiently low" risk with "sufficiently high" coverage. The functional relation between risk and coverage is called the *risk coverage (RC) trade-off*. Generally, the user of a selective model would like to bound one measure (either risk or coverage) and then obtain the best model in terms of the other measure. The *RC curve* of a given model characterizes this trade-off on a risk/coverage plane thus describing its full spectrum.

A selective predictor is useful if its RC curve is "non trivial" in the sense that progressively smaller risk can be obtained with progressively smaller coverage. Thus, when constructing a selective classification or a prediction model it is imperative to examine its RC curve. One can consider theoretical bounds of the RC curve (as in [6]) or empirical ones as we do here. Interpolated RC curve can be obtained by selecting a number of coverage bounds at certain grid points of choice, and learning (and testing) a selective model aiming at achieving the best possible risk for each coverage level. Obviously, each such model should respect the corresponding coverage bound.

# 3 Selective Prediction with HMMs

## 3.1 Ambiguity Model

The first approach we consider is an implementation of the classical ambiguity idea. We construct an HMM-based classifier, similar to the one used in [3], and endow it with a rejection mechanism in the spirit of Chow [5]. This approach is limited to binary labeled observation sequences. The training set, consisting of labeled sequences, is partitioned into its positive and negative instances, and two HMM's, $\lambda^+$ and $\lambda^-$, are trained using those sets, respectively. Thus, $\lambda^+$ is trained to identify positively labeled sequences, and $\lambda^-$ – negatively labeled sequences. Then, each new observation sequence $O$ is classified as $\text{sign}(\mathbb{P}\left[O\,|\,\lambda^+\right] - \mathbb{P}\left[O\,|\,\lambda^-\right])$.

For applying Chow's ambiguity idea using the model $(\lambda^+, \lambda^-)$, we need to define a measure $C(O)$ of prediction confidence for any observation sequence $O$. A natural choice in this context is to measure the log-likelihood difference between the positive and negative models, normalized by the length of the sequence. Thus, we define $C(O) \triangleq |\frac{1}{T}(\log \mathbb{P}\left[O\,|\,\lambda^+\right] - \log \mathbb{P}\left[O\,|\,\lambda^-\right])|$, where $T$ is the length of $O$. The greater $C(O)$ is, the more confident are we in the classification of $O$. Now, given the classification confidences of all sequences in the training data set, and given a required lower bound on the coverage, an empirical threshold can be found such that a designated number of instances with the smallest confidence measures will be rejected. If our data is non-stationary (e.g. financial sequences), this threshold can be re-estimated at the arrival of every new data instance.

## 3.2 State-Based Selectivity

We propose a different approach for implementing selective prediction with HMMs. The idea is to designate an appropriate subset of the states as "*rejective*." The proposed approach is suitable for prediction problems whose observation sequences are labeled. Specifically, for each observation, $O_t$, we assume that there is a corresponding label $l_t$. The goal is to predict $l_t$ at time $t-1$.

Each state is assigned *risk* and *visit rate* estimates. For each state $q$, its risk estimate is used as a proxy to the probability of making erroneous predictions from $q$, and its visit rate quantifies the probability of outputting any symbol from $q$. A subset of the highest risk states is selected so that their total expected visit rate does not exceed the user specified rejection bound. These states are called *rejective* and predictions from them are ignored. The following two definitions formulate these notions. We associate with each state $q$ a label $L_q$ representing the HMM prediction while at this state (see Section 3.4). Denote $\gamma_t(i) \triangleq \mathbb{P}\left[S_t = q_i\,|\,O, \lambda\right]$, and note that $\gamma_t(i)$ can be efficiently calculated using the standard forward-backward procedure (see Rabiner [16]).

**Definition 3.1 (emprirical visit rate).** Given an observation sequence, the empirical *visit rate*, $v(i)$, of a state $q_i$, is the fraction of time the HMM spends in state $q_i$, that is $v(i) \triangleq \frac{1}{T} \sum_{t=1}^{T} \gamma_t(i)$.

**Definition 3.2 (empirical state risk).** Given an observation sequence, the empirical *risk*, $r(i)$, of a state $q_i$, is the rate of erroneous visits to $q_i$, that is $r(i) \triangleq \frac{1}{v(i)T} \sum_{\substack{t=1 \\ L_{q_i} \neq l_t}}^{T} \gamma_t(i)$.

Suppose we are required to meet a user specified rejection bound $0 \leq B \leq 1$. This means that we are required to emit predictions (rather than "don't know"s) in at least $1 - B$ fraction of the time. To achieve this we apply the following greedy selection procedure of rejective states whereby highest risk states are sequentially selected as long as their overall visit rate does not exceed $B$. We call the resulting model *Naive-sHMM*. Formally, let $q_{i_1}, q_{i_2}, \ldots, q_{i_N}$ be an ordering of all states, such that for each $j < k$, $r(i_j) \geq r(i_k)$. Then, the rejective state subset is,

$$RS \triangleq \left\{ q_{i_1}, \ldots, q_{i_K} \;\middle|\; \sum_{j=1}^{K} v(i_j) \leq B, \sum_{j=1}^{K+1} v(i_j) > B \right\}. \tag{3.1}$$

## 3.3 Overcoming Coarseness

The above simple approach suffers from the following *coarseness* problem. If our model does not include a large number of states, or includes states with very high visit rates (as it is often the case in applications), the total visit rate of the rejective states might be far from the requested bound

$B$, entailing that selectivity cannot be fully exploited. For example, consider a model that has three states such that $r(q_1) > r(q_2) > r(q_3)$, $v(q_1) = \epsilon$, and $v(q_2) = B + \epsilon$. In this case only the negligibly visited $q_1$ will be rejected. We propose two methods to overcome this coarseness problem. These methods are presented in the two subsequent sections.

### 3.3.1 Randomized Linear Interpolation (RLI)

In the *randomized linear interpolation (RLI)* method, predictions from rejective states are always rejected, but predictions from the non-rejective state with the highest risk rate are rejected with appropriate probability, such that the total expected rejection rate equals the rejection bound $B$. Let $q$ be the non-rejective state with the highest risk rate. The probability to reject predictions emerging from this state is taken to be $p_q \triangleq \frac{1}{v(q)}\left(B - \sum_{q' \in RS} v(q')\right)$. Clearly, with $p_q$ thus defined, the total *expected* rejection rate is precisely $B$, when expectation is taken over random choices.

### 3.3.2 Recursive Refinement (RR)

Given an initial HMM model, the idea in the *recursive refinement* approach is to construct an approximate HMM whose states have finer granularity of visit rates. This smaller granularity enables a selection of rejective states whose total visit rate is closer to the required bound. The refinement is achieved by replacing every highly visited state with a complete HMM.

The process starts with a root HMM, $\lambda_0$, trained in a standard way using the Baum-Welch algorithm. In $\lambda_0$, states that have visit rate greater than a certain bound are identified. For each such state $q_i$ (called a *heavy* state), a new HMM $\lambda_i$ (called a *refining HMM*) is trained and combined with $\lambda_0$ as follows: every transition from other states into $q_i$ in $\lambda_0$ entails a transition into (initial) state in $\lambda_i$ in accordance with the initial state distribution of $\lambda_i$; every self transition to $q_i$ in $\lambda_0$ results in a state transition in $\lambda_i$ according to its state transition matrix; finally, every transition from $q_i$ to another state entails transition from a state in $\lambda_i$ whose probability is the original transition probability from $q_i$. States of $\lambda_i$ are assigned the label of $q_i$. This refinement continues in a recursive manner and terminates when all the heavy states have refinements. The non refined states are called *leaf* states.

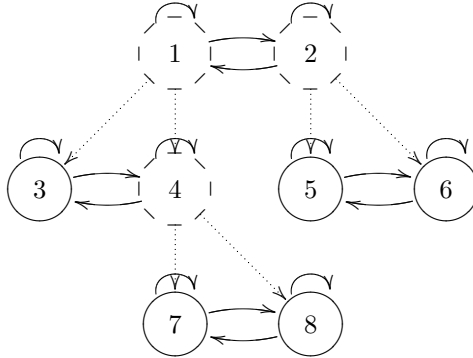

Figure 1: Recursively Refined HMM

Figure 1 depicts a recursively refined HMM having two refinement levels. In this model, states 1,2,4 are heavy (and refined) states, and states 3,5,6,7,8 are leaf (emitting) states. The model consisting of states 3 and 4 refines state 1, the model consisting of states 5 and 6 refines state 2, etc.

An *aggregate state* of the complete hierarchical model corresponds to a set of inner HMM states, each of which is a state on a path from the root through refining HMMs, to a leaf state. Only leaf states actually emit symbols. Refined states are non-emitting and their role in this construction is to preserve the structure (and transitions) of the HMMs they refine.

At every time instance $t$, the model is at some aggregate state. Transition to the next aggregate state always starts at $\lambda_0$, and recursively progresses to the leaf states, as shown in the following example. Suppose that the model in Figure 1 is at aggregate state $\{1,4,7\}$ at time $t$. The aggregate state at time $t+1$ is calculated as follows. $\lambda_0$ is in state 1, so its next state (say 1 again) is chosen according to the distribution $\{a_{11}, a_{12}\}$. We then consider the model that refines state 1, which was in state 4 at

time $t$. Here again the next state (say 3) is chosen according to the distribution $\{a_{43}, a_{44}\}$. State 3 is a leaf state that emits observations, and the aggregate state at time $t + 1$, is $\{1,3\}$. On the other hand, if state 2 is chosen at the root, a new state (say 6) in its refining model is chosen according to the initial distribution $\{\pi_5, \pi_6\}$ (transition into the heavy state from another state). The chosen state 6 is a leaf state so the new aggregate state becomes $\{2,6\}$.

---

**Algorithm 1** TrainRefiningHMM

---

**Input:** HMM $\lambda = \langle \{q_j\}_{j=1}^{j=n}, M, \pi, A, B \rangle$, heavy state $q_i$, $O$
1: Draw random HMM, $\lambda_i = \langle \{q_j\}_{j=n+1}^{j=n+N}, M, \{\pi_j\}_{j=n+1}^{j=n+N}, \{a_{jk}\}_{j,k=n+1}^{j,k=n+N}, \{b_{jm}\}_{j=n+1,m=1}^{j=n+N,m=M} \rangle$
2: For each $1 \leq j \leq n, j \neq i$, replace transition $q_j q_i$ with $q_j q_{n+1} \dots q_j q_{n+N}$, and $q_i q_j$ with $q_{n+1} q_j \dots q_{n+N} q_j$
3: Remove state $q_i$ with the corresponding $\{b_{im}\}_{i=1}^{i=M}$ from $\lambda$, and record it as a state refined by $\lambda_i$. Set $L_{q_j} = L_{q_i}$ for each $n + 1 \leq j \leq n + N$
4: **while** not converged **do**
5:     For each $1 \leq j \leq n, j \neq i$, update $a_{j(n+k)} = a_{ji} \pi_{n+k}$, and $a_{(n+k)j} = a_{ij}, 1 \leq k \leq N$.
6:     For each $n + 1 \leq j \leq n + N$, update $\pi_j = \pi_i \pi_j$
7:     For each $n + 1 \leq j, k \leq n + N$, update $a_{jk} = a_{ii} a_{jk}$
8:     Re-estimate $\{\pi_j\}_{j=n+1}^{j=n+N}, \{a_{jk}\}_{j,k=n+1}^{j,k=n+N}, \{b_{jm}\}_{j=n+1,m=1}^{j=n+N,m=M}$, using Eq.(3.2)
9: **end while**
10: Perform steps 5-7
**Output:** HMM $\lambda$

---

Algorithm 1 is a pseudocode of the training algorithm for refining HMM $\lambda_i$, for a heavy state $q_i$. This algorithm is an extension of the Baum-Welch algorithm [2]. In steps 1-3, a random $\lambda_i$ is generated and connected to the HMM $\lambda$ instead of $q_i$. Steps 5-8 iteratively update the parameters of $\lambda_i$ until the Baum-Welch convergence criterion is met, and in step 10, $\lambda$ is updated with the final $\lambda_i$ parameters. Finally, in step 3 $q_i$ is stored as a state refined by $\lambda_i$, to preserve the hierarchical structure of the resulting model (essential for the selection mechanism). The algorithm is applied on heavy states until all states in the HMM have visit rates lower than a required bound.

$$\pi_j = \frac{1}{Z} \left( \gamma_1(j) + \sum_{t=1}^{T-1} \sum_{\substack{k=1 \\ k \neq i}}^{n} \xi_t(k,j) \right), \quad a_{jk} = \frac{\sum_{t=1}^{T-1} \xi_t(j,k)}{\sum_{l=n+1}^{n+N} \sum_{t=1}^{T-1} \xi_t(j,l)}, \quad b_{jm} = \frac{\sum_{\substack{t=1 \\ O_t=m}}^{T} \gamma_t(j)}{\sum_{t=1}^{T} \gamma_t(j)} \quad (3.2)$$

In Eq. (3.2), re-estimation formulas for the parameters of newly added states (Step 8) are presented, where $\xi_t(j,k) = \mathbb{P}[q_t = j, q_{t+1} = k \mid O, \lambda]$. It is easy to see that, similarly to original Baum-Welch formulas, constraints for the parameters to be valid distributions are preserved ($Z$ is a normalization factor in the $\pi_j$ equation). The main difference from the original formulas is in the re-estimation of $\pi_j$: in the refinement process, transitions from other states into heavy state $q_i$ also affect the initial distribution of its refining states.

The most likely aggregate state at time $t$, given sequence $O$, is found in a top-down manner using the hierarchical structure of the model. Starting with the root model, $\lambda_0$, the most likely individual state in it, say $q_i$, is identified. If this state has no refinement, then we are done. Otherwise, the most likely individual state in $\lambda_i$ (HMM that refines $q_i$), say $q_j$, is identified, and the aggregate state is updated to be $\{q_i, q_j\}$. The process continues until the last discovered state has no refinement.

The above procedure requires calculation of the quantity $\gamma_t(i)$ not only for the leaf states (where it is calculated using a standard forward-backward procedure), but also for the refined states. For those states, $\gamma_t(i) = \sum_{\substack{j=1 \\ q_j \text{ refines } q_i}}^{N} \gamma_t(j)$ is calculated recursively over the hierarchical structure.

The rejection subset is found using the Eq. (3.1), applied to the aggregate states of the refined model. Visit and risk estimates for the aggregate state $\{q_{i_1} \dots q_{i_k}\}$ are calculated using $\gamma_t(i_k)$, of a leaf state $q_{i_k}$ that identifies this aggregate state.

The outcome of the RR procedure is a tree of HMMs whose main purpose is to redistribute visit rates among states. This re-distribution is the key element that allows for achieving smooth RC curves. Various other hierarchical HMM schemes have been proposed in the literature [4, 8, 10, 18, 20]. While some of these schemes may appear similar to ours at first glance, they do not address the visit rate re-distribution objective. In fact, those models were developed to serve other purposes such as better modeling of sequences that have special structure (e.g., sequences hypothesized to be emerged from a hierarchical generative model).

### 3.4 State Labeling

It remains to address the assignment of labels to the states in our state-based selection models. Labels can be assigned to states a-priori, and then a supervised EM method can be used for training (this model is known as Class HMM), as in [15]. Alternatively, state labels can be calculated from the statistics of the states, if an unsupervised training method is used. In our setting, we are following the latter approach. For a state $q_i$, and given observation label $l$, we calculate the average number of visits (at $q_i$) whose corresponding label is $l$, as $\mathbb{E}\left[S_t = q_i \,|\, l_t = l, O, \lambda\right] = \sum_{1 \leq t \leq T, l_t = l} \gamma_t(i)$. Thus, $L_{q_i}$ is chosen to be an $l$ that maximized this quantity.

## 4 Experimental Results

We compared empirically the four selection mechanisms presented in Section 3, namely, the ambiguity model and the Naive, RLI, and RR sHMMs. All methods were compared on a next-day trend prediction of the S&P500 index. This problem is known to be very difficult, and recent experimental work by Rao and Hong [17] assessed that although HMM succeeds to achieve some positive edge, the accuracy is near fifty-fifty (51.72%) when a pure price data is used.

For our prediction task, we took as observation sequence directions of the S&P500 price changes. Specifically, the direction $d_t$, at time $t$, is $d_t \triangleq \text{sign}(p_{t+1} - p_t)$, where $p_t$ are close prices. The state-based models were fed with the series of partial sequences $o_t \triangleq d_{t-\ell+1}, \ldots, d_t$. For the ambiguity model, the partial sequences $d_{t-\ell+1}, \ldots, d_t$ were used as a pool of observation sequences.

In a preliminary small experiment we observed the advantage of the state-based approach over the ambiguity model. In order to validate this, we tried to falsify this hypothesis by optimizing the hyper-parameters of the ambiguity model in hindsight.

For the state-based models we used a 5-state HMM, and predictions were made using the label of the most likely individual state. Such HMMs are hypothesized to be sufficiently expressive to model a small number of basic market conditions such as strong/weak trends (up and down) and sideways markets [17, 23]. We have not tried to optimize this basic architecture and better results with more expressive models can be possibly achieved. For the ambiguity model we constructed two 8-state HMMs, where the length of a single observation sequence ($\ell$) is 5. This architecture was optimized in hindsight among all possibilities of up to 10 states, and up to length 8, for a single observation sequence. Every refining model in the RR procedure had the same structure, and the upper bound on the visit rate was fixed at 0.1. For sHMMs, the hyper-parameter $\ell$ was arbitrarily set to 3 (there is possibly room for further improvement by optimizing the model w.r.t. this hyper-parameter).

RC curves were computed for each technique by taking the linear grid of rejection rate bounds from 0 to 0.9 in steps of 0.1. For each bound, every model was trained and tested using 30-fold cross-validation, with each fold consisting of 10 random restarts. Test performance was measured by mean error rate, taken over the 30 folds, and standard error of the mean (SEM) statistics were also calculated to monitor statistical significance.

Since the price sequences we deal with are highly non-stationary, we employed a *walk-forward* scheme in which the model is trained over the window of past $W_p$ returns and then tested on the subsequent window of $W_f$ "future" returns. Then, we "walk forward" $W_f$ steps (days) in the return sequence (so that the next training segment ends where the last test segment ended) and the process repeats until we consume the entire data sequence. In the experiments we set $W_p = 2000$ and $W_f = 50$ (that is, in each step we learn to predict the next business quarter, day by day). The data sequence in this experiment consisted of the 3000 S&P500 returns from 1/27/1999 to 12/31/2010. With our walk forward procedure, the first 2000 points were only used for training the first model.

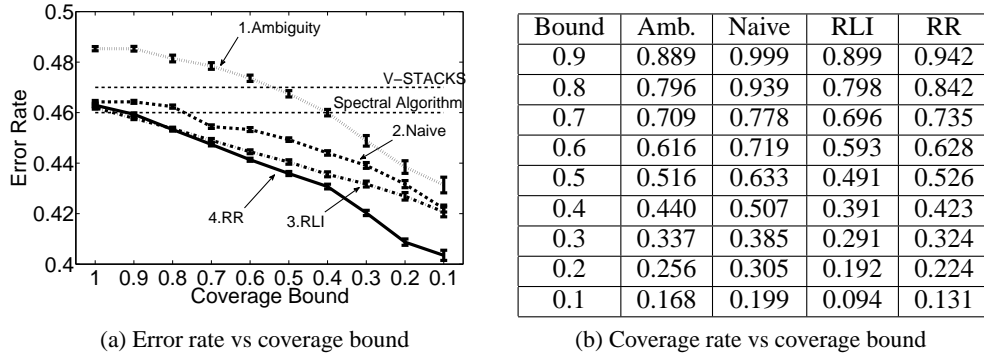

| Bound | Amb. | Naive | RLI | RR |
|-------|------|-------|-----|-----|
| 0.9 | 0.889 | 0.999 | 0.899 | 0.942 |
| 0.8 | 0.796 | 0.939 | 0.798 | 0.842 |
| 0.7 | 0.709 | 0.778 | 0.696 | 0.735 |
| 0.6 | 0.616 | 0.719 | 0.593 | 0.628 |
| 0.5 | 0.516 | 0.633 | 0.491 | 0.526 |
| 0.4 | 0.440 | 0.507 | 0.391 | 0.423 |
| 0.3 | 0.337 | 0.385 | 0.291 | 0.324 |
| 0.2 | 0.256 | 0.305 | 0.192 | 0.224 |
| 0.1 | 0.168 | 0.199 | 0.094 | 0.131 |

(a) Error rate vs coverage bound      (b) Coverage rate vs coverage bound

Figure 2: S&P500 RC-curves

Figure 2a shows that all four methods exhibited meaningful RC-curves; namely, the error rates decreased monotonically with decreasing coverage bounds. The RLI and RR models (curves 3 and 4, respectively) outperformed the Naive one (curve 2), by better exploiting the allotted coverage bound, as is evident from Table 2b. In addition, the RR model outperformed the RLI model, and moreover, its effective coverage is higher for every required coverage bound. This validates the effectiveness of the RR approach that implements a smarter selection process than the RLI model. Specifically, when RR refines a state and the resulting sub-states have different risk rates, the selection procedure will tend to reject riskier states first. Comparing the state-based models (curves 2-4) to the ambiguity model (curve 1), we see that all the state-based models outperformed the ambiguity model through the entire coverage range (despite the advantage we provided to the ambiguity model).

We also compared our models to two alternative HMM learning methods that were recently proposed: the spectral algorithm of Hsu et al. [13], and the V-STACKS algorithm of Siddiqi et al. [20]. As can be seen in Figure 2a, the selective techniques can also improve the accuracy obtained by these methods (with full coverage).

Quantitatively very similar results were also obtained in a number of other experiments (not presented, due to lack of space) with continuous data (without discretization) of the S&P500 index and of Gold, represented by its GLD exchange traded fund (ETF) replica.

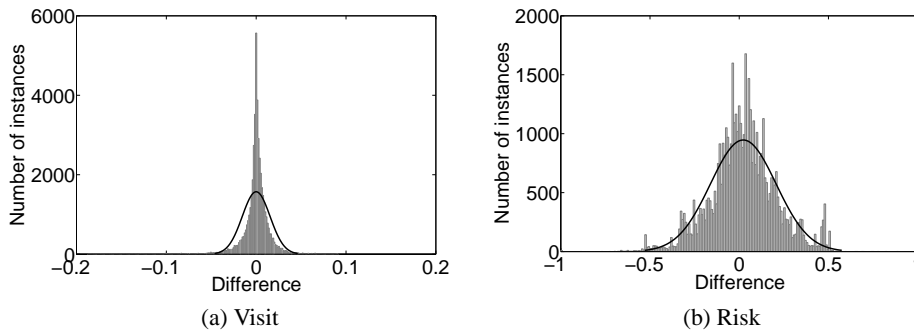

(a) Visit               (b) Risk

Figure 3: Distributions of visit and risk train/test differences

Figure 3a depicts the distribution of differences between empirical visit rates, measured on the training set, and those rates on the test set. It is evident that this distribution is symmetric and concentrated around zero. This means that our empirical visit estimates are quite robust and useful. Figure 3b depicts a similar distribution, but now for state risks. Unfortunately, here the distribution is much less concentrated, which means that our naive empirical risk estimates are rather noisy. While the distribution is symmetric about zero (and underestimates are often compensated by overestimates) it indicates that these noisy measurements are a major bottleneck in achieving better error rates. Therefore, it would be very interesting to consider more sophisticated risk estimation methods.

# 5 Related Work

Selective classification was introduced by Chow [5], who took a Bayesian route to infer the optimal rejection rule and analyze the risk-coverage trade-off under complete knowledge of the underlying probabilistic source. Chow's Bayes-optimal policy is to reject instances whenever none of the posteriori probabilities are sufficiently predominant. While this policy cannot be explicitly applied in agnostic settings, it marked a general *ambiguity-based* approach for rejection strategies. There is a substantial volume of research contributions on selective classification where the main theme is the implementation of reject mechanisms for particular classifier learning algorithms like support vector machines, see, e.g., [21]. Most of these mechanisms can be viewed as variations of the Chow ambiguity-based policy. The general consensus is that selective classification can often provide substantial error reductions and therefore rejection techniques have found good use in numerous applications, see, e.g., [12]. Rejection mechanisms were also utilized in [14] as a post-processing output verifier for HMM-based recognition systems There have been also a few theoretical studies providing worst case high probability bounds on the risk-coverage trade-off; see, .e.g., [1, 6, 7, 9].

HMMs have been extensively studied and used both theoretically and in numerous application areas. In particular, financial modeling with HMMs has been considered since their introduction by Baum et al. While a complete survey is clearly beyond our scope here, we mention a few related results. Hamilton [11] introduced a *regime-switching* model, in which the sequence is hypothesized to be generated by a number of hidden sources, or *regimes*, whose switching process is modeled by a (first-order) Markov chain. Later, in [19] a hidden Markov model of neural network "experts" was used for prediction of half-hour and daily price changes of the S&P500 index. Zhang [23] applied this model for predicting S&P500 next day trends, employing mixture of Gaussians in the states. The latter two works reported on prominent results in terms of cumulative profit. The recent experimental work by Rao and Hong [17] evaluated HMMs for a next-day trend prediction task and measured performance in terms of accuracy. They reported on a slight but consistent positive prediction edge.

In [3], an HMM-based classifier was proposed for "reliable trends," defined to be specialized 15 day return sequences that end with either five consecutive positive or consecutive negative returns. A classifier was constructed using two HMMs, one trained to identify upward (reliable) trends and the other, for downward (reliable) trends. Non-reliable sequences are always rejected. Therefore, this technique falls within selective prediction but the selection function has been manually predefined.

# 6 Concluding Remarks

The structure and modularity of HMMs make them particularly convenient for incorporating selective prediction mechanisms. Indeed, the proposed state-based method can result in a smooth and monotonically decreasing risk-coverage trade-off curve that allows for some control on the desired level of selectivity. We focused on selective prediction of trends in financial sequences. For these difficult prediction tasks our models can provide non-trivial prediction improvements. We expect that the relative advantage of these selective prediction techniques will be higher in easier tasks, or even in the same task by utilizing more elaborate HMM modeling, perhaps including other sources of specialized information including prices of other correlated indices.

We believe that a major bottleneck in attaining smaller test errors is the noisy risk estimates we obtain for the hidden states (see Figure 3b). This noise is partly due to the noisy nature of our prediction problem, but may also be attributed to the simplistic approach we took in estimating empirical risk. A challenging problem would be to incorporate more robust estimates in our mechanism, which is likely to enable better risk-coverage trade-offs. Finally, it will be very interesting to examine selective prediction mechanisms in the more general context of Bayesian networks and other types of graphical models.

### Acknowledgements

This work was supported in part by the IST Programme of the European Community, under the PASCAL2 Network of Excellence, IST-2007-216886. This publication only reflects the authors' views.

# References

[1] P. L. Bartlett and M. H. Wegkamp. Classification with a reject option using a hinge loss. *Journal of Machine Learning Research*, 9:1823–1840, 2008.

[2] L. E. Baum, T. Petrie, G. Soules, and N. Weiss. A maximization technique occurring in the statistical analysis of probabilistic functions of markov chains. *The Annals of Mathematical Statistics*, 41(1):164–171, 1970.

[3] M. Bicego, E. Grosso, and E. Otranto. A Hidden Markov Model approach to classify and predict the sign of financial local trends. *SSPR*, 5342:852–861, 2008.

[4] M. Brand. Coupled Hidden Markov Models for modeling interacting processes. Technical Report 405, MIT Media Lab, 1997.

[5] C. Chow. On optimum recognition error and reject tradeoff. *IEEE-IT*, 16:41–46, 1970.

[6] R. El-Yaniv and Y. Wiener. On the foundations of noise-free selective classification. *JMLR*, 11:1605–1641, May 2010.

[7] R. El-Yaniv and Y. Wiener. Agnostic selective classification. In *NIPS*, 2011.

[8] S. Fine, Y. Singer, and N. Tishby. The Hierarchical Hidden Markov Model: Analysis and Applications. *Machine Learning*, 32(1):41–62, 1998.

[9] Y. Freund, Y. Mansour, and R. E. Schapire. Generalization bounds for averaged classifiers. *Annals of Statistics*, 32(4):1698–1722, 2004.

[10] Z. Ghahramani and M. I. Jordan. Factorial Hidden Markov Models. *Machine Learning*, 29(2–3):245–273, 1997.

[11] J. Hamilton. Analysis of time series subject to changes in regime. *Journal of Econometrics*, 45(1–2):39–70, 1990.

[12] B. Hanczar and E. R. Dougherty. Classification with reject option in gene expression data. *Bioinformatics*, 24:1889–1895, 2008.

[13] D. Hsu, S. Kakade, and T. Zhang. A spectral algorithm for learning Hidden Markov Models. In *COLT*, 2009.

[14] A. L. Koerich. Rejection strategies for handwritten word recognition. In *IWFHR*, 2004.

[15] A. Krogh. Hidden Markov Models for labeled sequences. In *Proceedings of the 12th IAPR ICPR'94*, pages 140–144, 1994.

[16] L. R. Rabiner. A tutorial on Hidden Markov Models and selected applications in speech recognition. *Proceedings of the IEEE*, 77(2), February 1989.

[17] S. Rao and J. Hong. Analysis of Hidden Markov Models and Support Vector Machines in financial applications. Technical Report UCB/EECS-2010-63, Electrical Engineering and Computer Sciences University of California at Berkeley, 2010.

[18] L. K. Saul and M. I. Jordan. Mixed memory Markov models: Decomposing complex stochastic processes as mixtures of simpler ones. *Machine Learning*, 37:75–87, 1999.

[19] S. Shi and A. S. Weigend. Taking time seriously: Hidden Markov Experts applied to financial engineering. In *IEEE/IAFE*, pages 244–252. IEEE, 1997.

[20] S. Siddiqi, G. Gordon, and A. Moore. Fast State Discovery for HMM Model Selection and Learning. In *AI-STATS*, 2007.

[21] F. Tortorella. Reducing the classification cost of support vector classifiers through an ROC-based reject rule. *Pattern Anal. Appl.*, 7:128–143, 2004.

[22] A. Viterbi. Error bounds for convolutional codes and an asymptotically optimum decoding algorithm. *IEEE-IT*, 13(2):260–269, 1967.

[23] Y. Zhang. Prediction of financial time series with Hidden Markov Models. Master's thesis, The School of Computing Science, Simon Frazer University, Canada, 2004.

